# The "tree-dependent components" of natural scenes are edge filters

**Daniel Zoran**
Interdisciplinary Center for Neural Computation
Hebrew University of Jerusalem
daniez@cs.huji.ac.il

**Yair Weiss**
School of Computer Science
Hebrew University of Jerusalem
yweiss@cs.huji.ac.il

## Abstract

We propose a new model for natural image statistics. Instead of minimizing dependency between components of natural images, we maximize a simple form of dependency in the form of tree-dependencies. By learning filters and tree structures which are best suited for natural images we observe that the resulting filters are edge filters, similar to the famous ICA on natural images results. Calculating the likelihood of an image patch using our model requires estimating the squared output of pairs of filters connected in the tree. We observe that after learning, these pairs of filters are predominantly of similar orientations but different phases, so their joint energy resembles models of complex cells.

## 1  Introduction and related work

Many models of natural image statistics have been proposed in recent years [1, 2, 3, 4]. A common goal of many of these models is finding a representation in which components or sub-components of the image are made as independent or as sparse as possible [5, 6, 2]. This has been found to be a difficult goal, as natural images have a highly intricate structure and removing dependencies between components is hard [7]. In this work we take a different approach, instead of *minimizing dependence* between components we try to *maximize* a simple form of *dependence* - tree dependence.

It would be useful to place this model in context of previous works about natural image statistics. Many earlier models are described by the marginal statistics solely, obtaining a factorial form of the likelihood:

$$p(x) = \prod_i p_i(x_i) \tag{1}$$

The most notable model of this approach is Independent Component Analysis (ICA), where one seeks to find a linear transformation which maximizes independence between components (thus fitting well with the aforementioned factorization). This model has been applied to many scenarios, and proved to be one of the great successes of natural image statistics modeling with the emergence of edge-filters [5]. This approach has two problems. The first is that dependencies between components are still very strong, even with those learned transformation seeking to remove them. Second, it has been shown that ICA achieves, after the learned transformation, only marginal gains when measured quantitatively against simpler method like PCA [7] in terms of redundancy reduction. A different approach was taken recently in the form of radial Gaussianization [8], in which components which are distributed in a radially symmetric manner are made independent by transforming them non-linearly into a radial Gaussian, and thus, independent from one another.

A more elaborate approach, related to ICA, is Independent Subspace Component Analysis or ISA. In this model, one looks for independent subspaces of the data, while allowing the sub-components

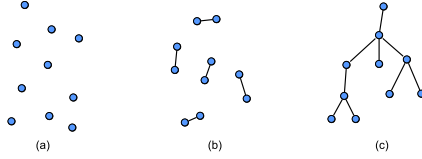

Figure 1: Our model with respect to marginal models such as ICA (a), and ISA like models (b). Our model, being a tree based model (c), allows components to belong to more than one subspace, and the subspaces are not required to be independent.

of each subspace to be dependent:

$$p(x) = \prod_k p_k(x_{i \in K}) \tag{2}$$

This model has been applied to natural images as well and has been shown to produce the emergence of phase invariant edge detectors, akin to complex cells in V1 [2].

Independent models have several shortcoming, but by far the most notable one is the fact that the resulting components are, in fact, highly dependent. First, dependency between the responses of ICA filters has been reported many times [2, 7]. Also, dependencies between ISA components has also been observed [9]. Given these robust *dependencies* between filter outputs, it is somewhat peculiar that in order to get simple cell properties one needs to assume *independence*. In this work we ask whether it is possible to obtain V1 like filters in a model that assumes dependence.

In our model we assume the filter distribution can be described by a tree graphical model [10] (see Figure 1). Degenerate cases of tree graphical models include ICA (in which no edges are present) and ISA (in which edges are only present within a subspace). But in its non-degenerate form, our model assumes any two filter outputs may be dependent. We allow components to belong to more than one subspace, and as a result, do not *require* independence between them.

## 2   Model and learning

Our model is comprised of three main components. Given a set of patches, we look for the parameters which maximize the likelihood of a whitened natural image patch $\mathbf{z}$:

$$p(\mathbf{z}; \mathbf{W}, \beta, T) = p(y_1) \prod_{i=1}^{N} p(y_i | y_{pa_i}; \beta) \tag{3}$$

Where $\mathbf{y} = \mathbf{W}\mathbf{z}$, $T$ is the tree structure, $pa_i$ denotes the parent of node $i$ and $\beta$ is a parameter of the density model (see below for the details). The three components we are trying to learn are:

1. The filter matrix $\mathbf{W}$, where every row defines one of the filters. The response of these filters is assumed to be tree-dependent. We assume that W is orthogonal (and is a rotation of a whitening transform).

2. The tree structure $T$ which specifies which components are dependent on each other.

3. The probability density function for connected nodes in the tree, which specify the exact form of dependency between nodes.

All three together describe a complete model for whitened natural image patches, allowing likelihood estimation and exact inference [11].

We perform the learning in an iterative manner: we start by learning the tree structure and density model from the entire data set, then, keeping the structure and density constant, we learn the filters via gradient ascent in mini-batches. Going back to the tree structure we repeat the process many times iteratively. It is important to note that both the filter set and tree structure are learned from the data, and are continuously updated during learning. In the following sections we will provide details on the specifics of each part of the model.

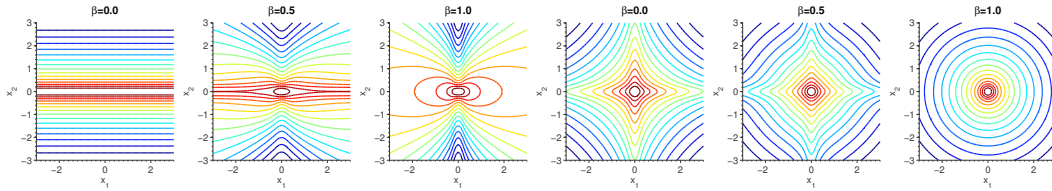

Figure 2: Shape of the conditional (Left three plots) and joint (Right three plots) density model in log scale for several values of $\beta$, from dependence to independence.

## 2.1 Learning tree structure

In their seminal paper, Chow and Liu showed how to learn the optimal tree structure approximation for a multidimensional probability density function [12]. This algorithm is easy to apply to this scenario, and requires just a few simple steps. First, given the current estimate for the filter matrix $\mathbf{W}$, we calculate the response of each of the filters with all the patches in the data set. Using these responses, we calculate the mutual information between each pair of filters (nodes) to obtain a fully connected weighted graph. The final step is to find a maximal spanning tree over this graph. The resulting unrooted tree is the optimal tree approximation of the joint distribution function over all nodes. We will note that the tree is *unrooted*, and the root can be chosen arbitrarily - this means that there is no node, or filter, that is more important than the others - the direction in the tree graph is arbitrary as long as it is chosen in a consistent way.

## 2.2 Joint probability density functions

Gabor filter responses on natural images exhibit highly kurtotic marginal distributions, with heavy tails and sharp peaks [13, 3, 14]. Joint pair wise distributions also exhibit this same shape with varying degrees of dependency between the components [13, 2]. The density model we use allows us to capture both the highly kurtotic nature of the distributions, while still allowing varying degrees of dependence using a mixing variable. We use a mix of two forms of finite, zero mean Gaussian Scale Mixtures (GSM). In one, the components are assumed to be independent of each other and in the other, they are assumed to be spherically distributed. The mixing variable linearly interpolates between the two, allowing us to capture the whole range of dependencies:

$$p(x_1, x_2; \beta) = \beta p_{dep}(x_1, x_2) + (1 - \beta)p_{ind}(x_1, x_2) \tag{4}$$

When $\beta = 1$ the two components are dependent (unless $p$ is Gaussian), whereas when $\beta = 0$ the two components are independent. For the density functions themselves, we use a finite GSM. The dependent case is a scale mixture of bivariate Gaussians:

$$p_{dep}(x_1, x_2) = \sum_k \pi_k \mathcal{N}(x_1, x_2; \sigma_k^2 \mathbf{I}) \tag{5}$$

While the independent case is a product of two independent univariate Gaussians:

$$p_{ind}(x_1, x_2) = \left( \sum_k \pi_k \mathcal{N}(x_1; \sigma_k^2) \right) \left( \sum_k \pi_k \mathcal{N}(x_2; \sigma_k^2) \right) \tag{6}$$

Estimating parameters $\pi_k$ and $\sigma_k^2$ for the GSM is done directly from the data using Expectation Maximization. These parameters are the same for all edges and are estimated only once on the first iteration. See Figure 2 for a visualization of the conditional distribution functions for varying values of $\beta$. We will note that the marginal distributions for the two types of joint distributions above are the same. The mixing parameter $\beta$ is also estimated using EM, but this is done for each edge in the tree separately, thus allowing our model to theoretically capture the fully independent case (ICA) and other degenerate models such as ISA.

## 2.3 Learning tree dependent components

Given the current tree structure and density model, we can now learn the matrix $\mathbf{W}$ via gradient ascent on the log likelihood of the model. All learning is performed on whitened, dimensionally

reduced patches. This means that $\mathbf{W}$ is a $N \times N$ rotation (orthonormal) matrix, where $N$ is the number of dimensions after dimensionality reduction (see details below). Given an image patch $\mathbf{z}$ we multiply it by $\mathbf{W}$ to get the response vector $\mathbf{y}$:

$$\mathbf{y} = \mathbf{W}\mathbf{z} \tag{7}$$

Now we can calculate the log likelihood of the given patch using the tree model (which we assume is constant at the moment):

$$\log p(\mathbf{y}) = \log p(y_{root}) + \sum_{i=1}^{N} \log p(y_i | y_{pa_i}) \tag{8}$$

Where $pa_i$ denotes the parent of node $i$. Now, taking the derivative w.r.t the $r$-th row of $\mathbf{W}$:

$$\frac{\partial \log p(\mathbf{y})}{\partial \mathbf{W}_r} = \frac{\partial \log p(\mathbf{y})}{\partial y_r} \mathbf{z}^T \tag{9}$$

Where $\mathbf{z}$ is the whitened natural image patch. Finally, we can calculate the derivative of the log likelihood with respect to the $r$-th element in $\mathbf{y}$:

$$\frac{\partial \log p(\mathbf{y})}{\partial y_r} = \frac{\partial \log p(y_{\mathbf{pa}_r}, y_r)}{\partial y_r} + \sum_{c \in C(r)} \frac{\partial \log p(y_r, y_c)}{\partial y_r} - \frac{\partial \log p(y_r)}{\partial y_r} \tag{10}$$

Where $C(r)$ denote the children of node $r$. In summary, the gradient ascent rule for updating the rotation matrix $\mathbf{W}$ is given by:

$$\mathbf{W}_r^{t+1} = \mathbf{W}_r^t + \eta \frac{\partial \log p(\mathbf{y})}{\partial y_r} \mathbf{z}^T \tag{11}$$

Where $\eta$ is the learning rate constant. After update, the rows of $\mathbf{W}$ are orthonormalized.

This gradient ascent rule is applied for several hundreds of patches (see details below), after which the tree structure is learned again as described in Section 2.1, using the new filter matrix $\mathbf{W}$, repeating this process for many iterations.

## 3 Results and analysis

### 3.1 Validation

Before running the full algorithm on natural image data, we wanted to validate that it does produce sensible results with simple synthetic data. We generated noise from four different models, one is $1/f$ independent Gaussian noise with 8 Discrete Cosine Transform (DCT) filters, the second is a simple ICA model with 8 DCT filters, and highly kurtotic marginals. The third was a simple ISA model - 4 subspaces, each with two filters from the DCT filter set. Distribution within the subspace was a circular, highly kurtotic GSM, and the subspaces were sampled independently. Finally, we generated data from a simple synthetic tree of DCT filters, using the same joint distributions as for the ISA model. These four synthetic random data sets were given to the algorithm - results can be seen in Figure 3 for the ICA, ISA and tree samples. In all cases the model learned the filters and distribution correctly, reproducing both the filters (up to rotations within the subspace in ISA) and the dependency structure between the different filters. In the case of $1/f$ Gaussian noise, any whitening transformation is equally likely and any value of beta is equally likely. Thus in this case, the algorithm cannot find the tree or the filters.

### 3.2 Learning from natural image patches

We then ran experiments with a set of natural images [9][1]. These images contain natural scenes such as mountains, fields and lakes. . The data set was 50,000 patches, each $16 \times 16$ pixels large. The patches' DC was removed and they were then whitened using PCA. Dimension was reduced from 256 to 128 dimensions. The GSM for the density model had 16 components. Several initial

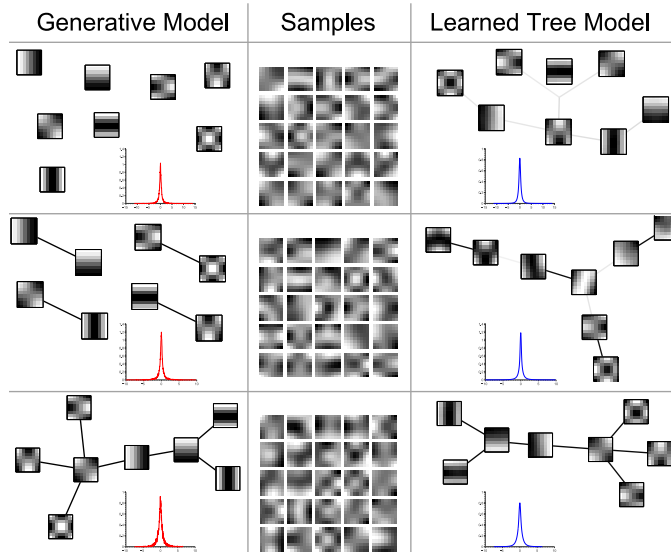

| Generative Model | Samples | Learned Tree Model |
|---|---|---|

Figure 3: Validation of the algorithm. Noise was generated from three models - top row is ICA, middle row is ISA and bottom row is a tree model. Samples were then given to the algorithm. On the right are the resulting learned tree models. Presented are the learned filters, tree model (with white edges meaning $\beta = 0$, black meaning $\beta = 1$ and grays intermediate values) and an example of a marginal histogram for one of the filters. It can be seen that in all cases all parts of the model were correctly learned. Filters in the ISA case were learned up to rotation within the subspace, and all filters were learned up to sign. $\beta$ values for the ICA case were always below 0.1, as were the values of $\beta$ between subspaces in ISA.

conditions for the matrix $\mathbf{W}$ were tried out (random rotations, identity) but this had little effect on results. Mini-batches of 10 patches each were used for the gradient ascent - the gradient of 10 patches was summed, and then normalized to have unit norm. The learning rate constant $\eta$ was set to 0.1. Tree structure learning and estimation of the mixing variable $\beta$ were done every 500 mini-batches. All in all, 50 iterations were done over the data set.

### 3.3 Filters and tree structure

Figures 4 and 5 show the learned filters ($\mathbf{WQ}$ where $\mathbf{Q}$ is the whitening matrix) and tree structure ($T$) learned from natural images. Unlike the ISA toy data in figure 3, here a full tree was learned and $\beta$ is approximately one for all edges. The GSM that was learned for the marginals was highly kurtotic.

It can be seen that resulting filters are edge filters at varying scales, positions and orientations. This is similar to the result one gets when applying ICA to natural images [5, 15]. More interesting is

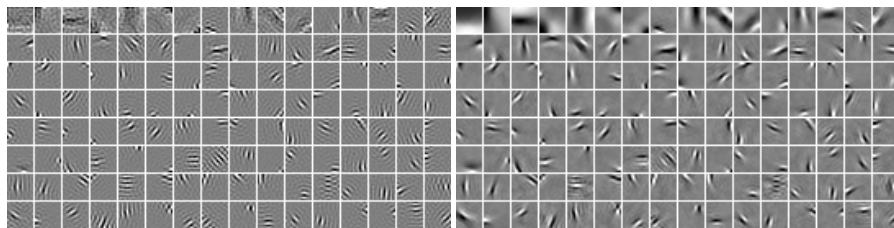

Figure 4: Left: Filter set learned from $16 \times 16$ natural image patches. Filters are ordered by PCA eigenvalues, largest to smallest. Resulting filters are edge filters having different orientations, positions, frequencies and phases. Right: The "feature" set learned, that is, columns of the pseudo inverse of the filter set.

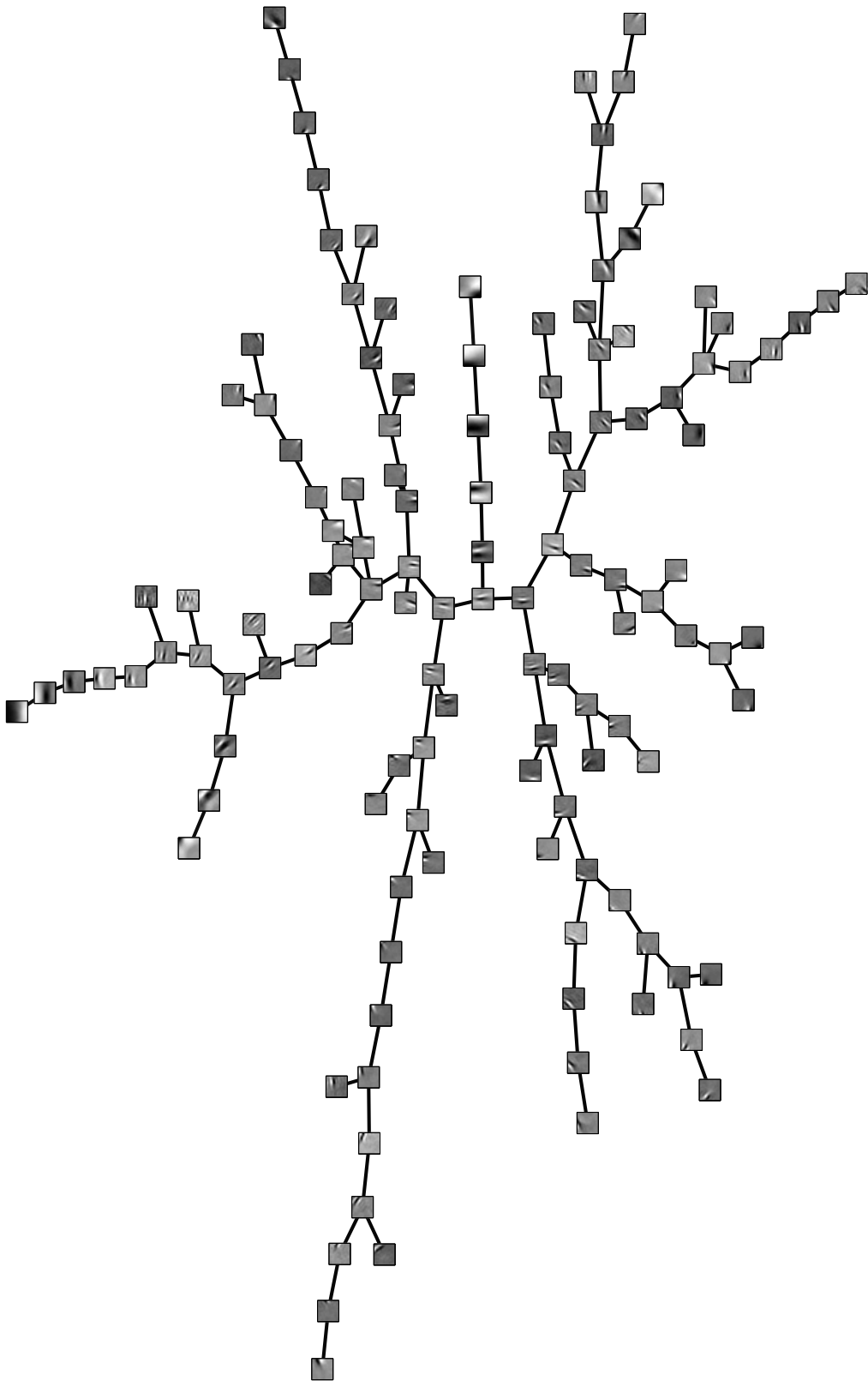

Figure 5: The learned tree graph structure and feature set. It can be seen that neighboring features on the graph have similar orientation, position and frequency. See Figure 4 for a better view of the feature details, and see text for full detail and analysis. Note that the figure is rotated CW.

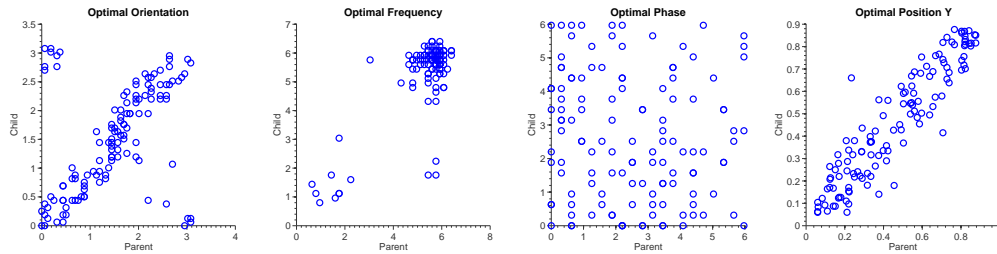

Figure 6: Correlation of optimal parameters in neighboring nodes in the tree graph. Orientation, frequency and position are highly correlated, while phase seems to be entirely uncorrelated. This property of correlation in frequency and orientation, while having no correlation in phase is related to the ubiquitous energy model of complex cells in V1. See text for further details.

| Model | Log Likelihood |
|---|---|
| Marginal PCA | -162.5 |
| Marginal ICA | -157.0 |
| ISA | -159.4 |
| Our Model | **-144.8** |

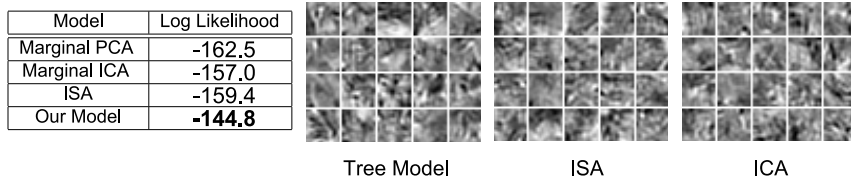

Tree Model        ISA        ICA

Figure 7: Left: Comparison of log likelihood values of our model with PCA, ICA and ISA. Our model gives the highest likelihood. Right: Samples taken at random from ICA, ISA and our model. Samples from our model appear to contain more long-range structure.

the tree graph structure learned along with the filters which is shown in Figure 5. It can be seen that neighboring filters (nodes) in the tree tend to have similar position, frequency and orientation. Figure 6 shows the correlation of optimal frequency, orientation and position for neighboring filters in the tree - it is obvious that all three are highly correlated. Also apparent in this figure is the fact that the optimal phase for neighboring filters has no significant correlation. It has been suggested that filters which have the same orientation, frequency and position with different phase can be related to complex cells in V1 [2, 16].

### 3.4 Comparison to other models

Since our model is a generalization of both ICA and ISA we use it to learn both models. In order to learn ICA we used the exact same data set, but the tree had no edges and was not learned from the data (alternatively, we could have just set $\beta = 0$). For ISA we used a forest architecture of 2 node trees, setting $\beta = 1$ for all edges (which means a spherical symmetric distribution), no tree structure was learned. Both models produce edge filters similar to what we learn (and to those in [5, 15, 6]). The ISA model produces neighboring nodes with similar frequency and orientation, but different phase, as was reported in [2]. We also compare to a simple PCA whitening transform, using the same whitening transform and marginals as in the ICA case, but setting $\mathbf{W} = \mathbf{I}$.

We compare the likelihood each model gives for a test set of natural image patches, different from the one that was used in training. There were 50,000 patches in the test set, and we calculate the mean log likelihood over the entire set. The table in Figure 7 shows the result - as can be seen, our model performs better in likelihood terms than both ICA and ISA.

Using a tree model, as opposed to more complex graphical models, allows for easy sampling from the model. Figure 7 shows 20 random samples taken from our tree model along with samples from the ICA and ISA models. Note the elongated structures (e.g. in the bottom left sample) in the samples from the tree model, and compare to patches sampled from the ICA and ISA models.

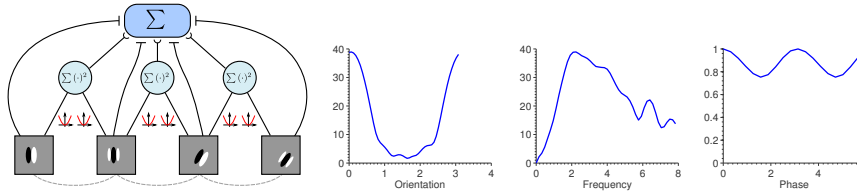

Figure 8: Left: Interpretation of the model. Given a patch, the response of all edge filters is computed ("simple cells"), then at each edge, the corresponding nodes are squared and summed to produce the response of the "complex cell" this edge represents. Both the response of complex cells and simple cells is summed to produce the likelihood of the patch. Right: Response of a "complex cell" in our model to changing phase, frequency and orientation. Response in the y-axis is the sum of squares of the two filters in this "complex cell". Note that while the cell is selective to orientation and frequency, it is rather invariant to phase.

## 3.5   Tree models and complex cells

One way to interpret the model is looking at the likelihood of a given patch under this model. For the case of $\beta = 1$ substituting Equation 4 into Equation 3 yields:

$$\log L(\mathbf{z}) = \sum_i \rho(\sqrt{y_i^2 + y_{pa_i}^2}) - \rho(|y_{pa_i}|) \tag{12}$$

Where $\rho(x) = \log\left(\sum_k \pi_k \mathcal{N}(x; \sigma_k^2)\right)$. This form of likelihood has an interesting similarity to models of complex cells in V1 [2, 4]. In Figure 8 we draw a simple two-layer network that computes the likelihood. The first layer applies linear filters ("simple cells") to the image patch, while the second layer sums the squared outputs of similarly oriented filters from the first layer, having different phases, which are connected in the tree ("complex cells"). Output is also dependent on the actual response of the "simple cell" layer. The likelihood here is maximized when both the response of the parent filter $y_{pa_i}$ and the child $y_i$ is zero, but, given that one filter has responded with a non-zero value, the likelihood is maximized when the other filter also fires (see the conditional density in Figure 2). Figure 8 also shows an example of the phase invariance which is present in the learned "complex cell" (energy of a pair of learned filters connected in the tree) - it seems that sum squared response of the shown pair of nodes is relatively invariant to the phase of the stimulus, while it is selective to both frequency and orientation - the hallmark of "complex cells". Quantifying this result with the AC/DC ratio, as is common [17] we find that around 60% percent of the edges have an AC/DC ratio which is smaller than one - meaning they would be classified as complex cells using standard methods [17].

## 4   Discussion

We have proposed a new model for natural image statistics which, instead of minimizing dependency between components, maximizes a simple form of dependency - tree dependency. This model is a generalization of both ICA and ISA. We suggest a method to learn such a model, including the tree structure, filter set and density model. When applied to natural image data, our model learns edge filters similar to those learned with ICA or ISA. The ordering in the tree, however, is interesting - neighboring filters in the tree tend to have similar orientation, position and frequency, but different phase. This decorrelation of phase, in conjunction with correlations in frequency and orientation are the hallmark of energy models for complex cells in V1.

Future work will include applications of the model to several image processing scenarios. We have started experimenting with application of this model to image denoising by using belief propagation for inference, and results are promising.

## Acknowledgments

This work has been supported by the AMN foundation and the ISF. The authors wish to thank the anonymous reviewers for their helpful comments.

## Footnotes

[1]available at http://www.cis.hut.fi/projects/ica/imageica/

# References

[1] Y. Weiss and W. Freeman, "What makes a good model of natural images?" *Computer Vision and Pattern Recognition, 2007. CVPR '07. IEEE Conference on*, pp. 1–8, June 2007.

[2] A. Hyvarinen and P. Hoyer, "Emergence of phase-and shift-invariant features by decomposition of natural images into independent feature subspaces," *Neural Computation*, vol. 12, no. 7, pp. 1705–1720, 2000.

[3] A. Srivastava, A. B. Lee, E. P. Simoncelli, and S.-C. Zhu, "On advances in statistical modeling of natural images," *J. Math. Imaging Vis.*, vol. 18, no. 1, pp. 17–33, 2003.

[4] Y. Karklin and M. Lewicki, "Emergence of complex cell properties by learning to generalize in natural scenes," *Nature*, November 2008.

[5] A. J. Bell and T. J. Sejnowski, "The independent components of natural scenes are edge filters," *Vision Research*, vol. 37, pp. 3327–3338, 1997.

[6] B. Olshausen *et al.*, "Emergence of simple-cell receptive field properties by learning a sparse code for natural images," *Nature*, vol. 381, no. 6583, pp. 607–609, 1996.

[7] M. Bethge, "Factorial coding of natural images: how effective are linear models in removing higher-order dependencies?" vol. 23, no. 6, pp. 1253–1268, June 2006.

[8] S. Lyu and E. P. Simoncelli, "Nonlinear extraction of 'independent components' of natural images using radial Gaussianization," *Neural Computation*, vol. 21, no. 6, pp. 1485–1519, Jun 2009.

[9] A. Hyvrinen, P. Hoyer, and M. Inki, "Topographic independent component analysis: Visualizing the dependence structure," in *Proc. 2nd Int. Workshop on Independent Component Analysis and Blind Signal Separation (ICA2000), Espoo, Finland*.   Citeseer, 2000, pp. 591–596.

[10] F. Bach and M. Jordan, "Beyond independent components: trees and clusters," *The Journal of Machine Learning Research*, vol. 4, pp. 1205–1233, 2003.

[11] J. Yedidia, W. Freeman, and Y. Weiss, "Understanding belief propagation and its generalizations," *Exploring artificial intelligence in the new millennium*, pp. 239–236, 2003.

[12] C. Chow and C. Liu, "Approximating discrete probability distributions with dependence trees," *IEEE transactions on Information Theory*, vol. 14, no. 3, pp. 462–467, 1968.

[13] E. Simoncelli, "Bayesian denoising of visual images in the wavelet domain," *LECTURE NOTES IN STATISTICS-NEW YORK-SPRINGER VERLAG-*, pp. 291–308, 1999.

[14] A. Levin, A. Zomet, and Y. Weiss, "Learning to perceive transparency from the statistics of natural scenes," *Advances in Neural Information Processing Systems*, pp. 1271–1278, 2003.

[15] J. van Hateren, "Independent component filters of natural images compared with simple cells in primary visual cortex," *Proceedings of the Royal Society B: Biological Sciences*, vol. 265, no. 1394, pp. 359–366, 1998.

[16] C. Zetzsche, E. Barth, and B. Wegmann, "The importance of intrinsically two-dimensional image features in biological vision and picture coding," in *Digital images and human vision*. MIT Press, 1993, p. 138.

[17] K. Kording, C. Kayser, W. Einhauser, and P. Konig, "How are complex cell properties adapted to the statistics of natural stimuli?" *Journal of Neurophysiology*, vol. 91, no. 1, pp. 206–212, 2004.

